# Density Propagation and
# Improved Bounds on the Partition Function*

**Stefano Ermon, Carla P. Gomes**
Dept. of Computer Science
Cornell University
Ithaca NY 14853, U.S.A.

**Ashish Sabharwal**
IBM Watson Research Ctr.
Yorktown Heights
NY 10598, U.S.A.

**Bart Selman**
Dept. of Computer Science
Cornell University
Ithaca NY 14853, U.S.A.

## Abstract

Given a probabilistic graphical model, its *density of states* is a distribution that, for any likelihood value, gives the number of configurations with that probability. We introduce a novel message-passing algorithm called Density Propagation (DP) for estimating this distribution. We show that DP is exact for tree-structured graphical models and is, in general, a strict generalization of both sum-product and max-product algorithms. Further, we use density of states and tree decomposition to introduce a new family of upper and lower bounds on the *partition function*. For any tree decomposition, the new upper bound based on finer-grained density of state information is provably at least as tight as previously known bounds based on convexity of the log-partition function, and strictly stronger if a general condition holds. We conclude with empirical evidence of improvement over convex relaxations and mean-field based bounds.

## 1 Introduction

Associated with any undirected graphical model [1] is the so-called density of states, a term borrowed from statistical physics indicating a distribution that, for any likelihood value, gives the number of configurations with that probability. The density of states plays an important role in statistical physics because it provides a fine grained description of the system, and can be used to efficiently compute many properties of interests, such as the partition function and its parameterized version [2, 3]. It can be seen that computing the density of states is computationally intractable in the worst case, since it subsumes a #-P complete problem (computing the partition function) and an NP-hard one (MAP inference). All current approximate techniques estimating the density of states are based on sampling, the most prominent being the Wang-Landau algorithm [3] and its improved variants [2]. These methods have been shown to be very effective in practice. However, they do not provide any guarantee on the quality of the results. Furthermore, they ignore the structure of the underlying graphical model, effectively treating the energy function (which is proportional to the negative log-likelihood of a configuration) as a black-box.

As a first step towards exploiting the structure of the graphical model when computing the density of states, we propose an algorithm called DENSITYPROPAGATION (DP). The algorithm is based on dynamic programming and can be conveniently expressed in terms of message passing on the graphical model. We show that DENSITYPROPAGATION computes the density of states exactly for any tree-structured graphical model. It is closely related to the popular Sum-Product (Belief Propagation, BP) and Max-Product (MP) algorithms, and can be seen as a generalization of both. However, it computes something much richer, namely the density of states, which contains information such as the partition function and variable marginals. Although we do not work at the level of individual configurations, DENSITYPROPAGATION allows us to reason in terms of groups of configurations with the same probability (energy).

Being able to solve inference tasks for certain tractable classes of problems (e.g., trees) is important because one can often decompose a complex problem into tractable subproblems (such as spanning trees) [4], and the solutions to these simpler problems can be combined to recover useful properties of the original graphical model [5, 6]. In this paper we show that by combining the additional information given by the density of states, we can obtain a new family of upper and lower bounds on the partition function. We prove that the new upper bound is always at least as tight as the one based on the convexity of the log-partition function [4], and we provide a general condition where the new bound is strictly tighter. Further, we illustrate empirically that the new upper bound improves upon the convexity-based one on Ising grid and clique models, and that the new lower bound is empirically slightly stronger than the one given by mean-field theory [4, 7].

## 2   Problem definition and setup

We consider a graphical model specified as a factor graph with $N = |V|$ discrete random variables $x_i, i \in V$ where $x_i \in \mathcal{X}_i$. The global random vector $x = \{x_s, s \in V\}$ takes value in the Cartesian product $\mathcal{X} = \mathcal{X}_1 \times \mathcal{X}_2 \times \cdots \times \mathcal{X}_N$, with cardinality $D = |\mathcal{X}| = \prod_{i=1}^{N} |X_i|$. We consider a probability distribution over elements $x \in \mathcal{X}$ (called configurations)

$$p(x) = \frac{1}{Z} \prod_{\alpha \in \mathcal{I}} \psi_\alpha(\{x\}_\alpha) \tag{1}$$

that factors into factors $\psi_\alpha : \{x\}_\alpha \to \mathbb{R}^+$, where $\mathcal{I}$ is an index set and $\{x\}_\alpha \subseteq V$ a subset of variables the factor $\psi_\alpha$ depends on, and $Z$ is a normalization constant known as partition function. The corresponding factor graph is a bipartite graph with vertex set $V \cup \mathcal{I}$. In the factor graph, each variable node $i \in V$ is connected with all the factors $\alpha \in \mathcal{I}$ that depend on $i$. Similarly, each factor node $\alpha \in \mathcal{I}$ is connected with all the variable nodes $i \in \{x\}_\alpha$. We denote the neighbors of $i$ and $\alpha$ by $\mathcal{N}(i)$ and $\mathcal{N}(\alpha)$ respectively.

We will also make use of the related exponential representation [8]. Let $\phi$ be a collection of potential functions $\{\phi_\alpha, \alpha \in \mathcal{I}\}$, defined over the index set $\mathcal{I}$. Given an exponential parameter vector $\Theta = \{\Theta_\alpha, \alpha \in \mathcal{I}\}$, the exponential family defined by $\phi$ is the family of probability distributions over $\mathcal{X}$ defined as follows:

$$p(x, \Theta) = \frac{1}{Z(\Theta)} \exp(\Theta \cdot \phi(x)) = \frac{1}{Z(\Theta)} \exp\left(\sum_{\alpha \in \mathcal{I}} \Theta_\alpha \phi_\alpha(\{x\}_\alpha)\right) \tag{2}$$

where we assume $p(x) = p(x, \Theta^*)$. Given an exponential family, we define the *density of states* [2] as the following distribution:

$$n(E, \Theta) = \sum_{x \in \mathcal{X}} \delta\left(E - \Theta \cdot \phi(x)\right) \tag{3}$$

where $\delta\left(E - \Theta \cdot \phi(x)\right)$ indicates a Dirac delta centered at $\Theta \cdot \phi(x)$. For any exponential parameter $\Theta$, it holds that

$$\int_{-\infty}^{A} n(E, \Theta) \mathrm{d}E = |\{x \in \mathcal{X} | \Theta \cdot \phi(x) \le A\}|$$

and $\int_{\mathbb{R}} n(E, \Theta) \mathrm{d}E = |\mathcal{X}|$. We will refer to the quantity $\sum_{\alpha \in \mathcal{I}} \Theta_\alpha^* \phi_\alpha(\{x\}_\alpha) = \sum_{\alpha \in \mathcal{I}} \log \psi_\alpha(\{x\}_\alpha)$ as the energy of a configuration $x$, although it has an additional minus sign with respect to the conventional energy in statistical physics.

## 3   Density Propagation

Since any propositional Satisfiability (SAT) instance can be efficiently encoded as a factor graph (e.g., by defining a uniform probability measure over satisfying assignments), it is clear that computing the density of states is computationally intractable in the worst case, as a generalization of an NP-Complete problem (satisfiability testing) and a #-P complete problem (model counting).

We show that the density of states can be computed efficiently[1] for acyclic graphical models. We provide a Dynamic Programming algorithm, which can also be interpreted as a message passing algorithm on the factor graph, called DENSITYPROPAGATION (DP), which computes the density of states exactly for acyclic graphical models.

## 3.1 Density propagation equations

DENSITYPROPAGATION works by exchanging messages from variable to factor nodes and vice versa. Unlike traditional message passing algorithms, where messages represent marginal probabilities (vectors of real numbers), for every $x_i \in \mathcal{X}_i$ a DENSITYPROPAGATION message $m_{a \to i}(x_i)$ is a distribution (a "marginal" density of states), i.e. $m_{a \to i}(x_i) = \sum_k c_k(a \to i, x_i) \delta_{E_k(a \to i, x_i)}$ is a sum of Dirac deltas.

At every iteration, messages are updated according to the following rules. The message from variable node $i$ to factor node $a$ is updated as follows:

$$m_{i \to a}(x_i) = \bigotimes_{b \in \mathcal{N}(i) \backslash a} m_{b \to i}(x_i) \tag{4}$$

where $\bigotimes$ is the convolution operator (commutative, associative and distributive). Intuitively, the convolution operation gives the distribution of the sum of (conditionally) independent random variables, in this case corresponding to distinct subtrees in a tree-structured graphical model. The message from factor $a$ to variable $i$ is updated as follows:

$$m_{a \to i}(x_i) = \sum_{\{x\}_{\alpha \backslash i}} \left( \bigotimes_{j \in \mathcal{N}(a) \backslash i} m_{j \to a}(x_j) \right) \bigotimes \delta_{E_\alpha(\{x\}_\alpha)} \tag{5}$$

where $\delta_{E_\alpha(\{x\}_\alpha)}$ is a Dirac delta function centered at $E_\alpha(x_\alpha) = \log \psi_\alpha(\{x\}_\alpha)$.

For tree structured graphical models, DENSITYPROPAGATION converges after a finite number of iterations, independent of the initial condition, to the true density of states. Formally,

**Theorem 1.** *For any variable $i \in V$ and $A \in \mathbb{R}$, for any initial condition, after a finite number of iterations $\left( \sum_{q \in \mathcal{X}_s} \bigotimes_{b \in \mathcal{N}(i)} m_{b \to i}(q) \right)(E) = n(E, \Theta^*)$.*

The proof is by induction on the size of the tree (omitted due to lack of space).

### 3.1.1 Complexity and Approximation with Energy Bins

The most efficient message update schedule for tree structured models is a two-pass procedure where messages are first sent from the leaves to the root node, and then propagated backwards from the root to the leaves. However, as with other message-passing algorithms, for tree structured instances the algorithm will converge with either a sequential or a parallel update schedule, with any initial condition for the messages. Although DP requires the same number of messages updates as BP and MP, DP updates are more expensive because they require the computation of convolutions. Specifically, each variable-to-factor update rule (4) requires $(N - 2)L$ convolutions, where $N$ is the number of neighbors of the variable node and $L$ is the number of states in the random variable. Each factor-to-variable update rule (5) requires summation over $N - 1$ variables, each of size $L$, requiring $O(L^N)$ convolutions. Using Fast Fourier Transform (FFT), each convolution takes $O(K \log K)$, where $K$ is the maximum number of non-zero entries in a message. In the worst case, the density of states can have an exponential number of non-zero entries (i.e., the finite number of possible energy values, which we will also refer to as "buckets"), for instance when potentials are set to logarithms of prime numbers, making every $x \in \mathcal{X}$ have a different probability. However, in many practical problems of interest (e.g., SAT/CSP models and certain grounded Markov Logic Networks [9]), the number of energy "buckets" is limited, e.g., bounded by the total number of constraints. For general graphical models, coarse-grain energy bins can be used, similar to the Wang-Landau algorithm [3], without losing much precision. Specifically, if we use bins of size $\epsilon/M$, where each bin corresponds to configurations with energy in the interval $[k\epsilon/M, (k + 1)\epsilon/M)$, the energy estimated for each configuration through $O(M)$ convolutions is at most an $O(\epsilon)$ *additive* value away from its true energy (as the quantization error introduced by energy binning is summed up across convolution steps). This also guarantees that the density of states with coarse-grain energy bins gives a constant factor approximation of the true partition function.

### 3.1.2 Relationship with sum and max product algorithms

DENSITYPROPAGATION is closely related to traditional message passing algorithms such as BP (Belief Propagation, Sum-Product) and MP (Max-Product), since it is based on the same (conditional) independence assumptions. Specifically, as shown by the next theorem, both BP and MP can

be seen as simplified versions of DENSITYPROPAGATION that consider only certain global statistics of the distributions represented by DENSITYPROPAGATION messages.

**Theorem 2.** *With the same initial condition and message update schedule, at every iteration we can recover Belief Propagation and Max-Product marginals from* DENSITYPROPAGATION *messages.*

*Proof.* Given a DP message $m_{i \to j}(x_j) = \sum_k c_k(i \to j, x_j) \delta_{E_k(i \to j, x_j)}$, the Max-Product algorithm corresponds to considering only the entry associated with the highest probability, i.e. $\gamma_{i \to j}(x_j) = f(m_{i \to j}(x_j)) \triangleq \max_k \{E_k(i \to j, x_j)\}$. According to DP updates in equations (4) and (5), the quantities $\gamma_{i \to j}(x_j)$ are updated as follows

$$\gamma_{i \to a}(x_i) = f\left(\bigotimes_{b \in \mathcal{N}(i) \setminus a} m_{b \to i}(x_i)\right) = \sum_{b \in \mathcal{N}(i) \setminus a} \gamma_{b \to i}(x_i)$$

$$\gamma_{a \to i}(x_i) = f\left(\sum_{\{x\}_{\alpha \setminus i}} \left(\bigotimes_{j \in \mathcal{N}(a) \setminus i} m_{j \to a}(x_j)\right) \bigotimes \delta_{E_\alpha(\{x\}_\alpha)}\right) = \max_{\{x\}_{\alpha \setminus i}} \sum_{j \in \mathcal{N}(a) \setminus i} \gamma_{j \to a}(x_j) + E_\alpha(\{x\}_\alpha)$$

These results show that the quantities $\gamma_{i \to j}(x_j)$ are updated according to the Max-Product algorithm (with messages in log-scale). To see the relationship with BP, for every DP message $m_{i \to j}(x_j)$, let us define

$$\mu_{i \to j}(x_j) = ||m_{i \to j}(x_j)(E) \exp(E)||_1 = \int_{\mathbb{R}} m_{i \to j}(x_j)(E) \exp(E) \mathrm{d}E$$

Notice that $\mu_{i \to j}(x_j)$ would correspond to an unnormalized marginal probability, assuming that $m_{i \to j}(x_j)$ is the density of states of the instance when variable $j$ is clamped to value $x_j$. According to DP updates in equation (4) and (5)

$$\mu_{i \to a}(x_i) = ||m_{i \to a}(x_i)(E) \exp(E)||_1 = \left|\left|\bigotimes_{b \in \mathcal{N}(i) \setminus a} m_{b \to i}(x_i)(E) \exp(E)\right|\right|_1 = \prod_{b \in \mathcal{N}(i) \setminus a} \mu_{b \to i}(x_i)$$

$$\mu_{a \to i}(x_i) = ||\mu_{a \to i}(x_i)(E) \exp(E)||_1 = \left|\left|\sum_{\{x\}_{\alpha \setminus i}} \left(\bigotimes_{j \in \mathcal{N}(a) \setminus i} m_{j \to a}(x_j)\right) \bigotimes \delta_{E_\alpha(\{x\}_\alpha)}(E) \exp(E)\right|\right|_1$$

$$= \sum_{\{x\}_{\alpha \setminus i}} \left|\left|\left(\bigotimes_{j \in \mathcal{N}(a) \setminus i} m_{j \to a}(x_j)\right) \bigotimes \delta_{E_\alpha(\{x\}_\alpha)}(E) \exp(E)\right|\right|_1 = \sum_{\{x\}_{\alpha \setminus i}} \psi_\alpha(\{x\}_\alpha) \prod_{j \in \mathcal{N}(a) \setminus i} \mu_{j \to a}(x_i)$$

that is we recover BP updates for the $\mu_{i \to j}$ quantities. Similarly, if we define temperature versions of the marginals $\mu_{i \to j}^T(x_j) \triangleq ||m_{i \to j}(x_j)(E) \exp(E/T)||_1$, we recover the temperature-versions of Belief Propagation updates, similar to [10] and [11]. □

As other message passing algorithms, DENSITYPROPAGATION updates are well defined also for loopy graphical models, even though there is no guarantee of convergence or correctness [12]. The correspondence with BP and MP (Theorem 2) however still holds: if loopy BP converges, then the corresponding quantities $\mu_{i \to j}$ computed from DP messages will converge as well, and to the same value (assuming the same initial condition and update schedule). Notice however that the convergence of the $\mu_{i \to j}$ does not imply the convergence of DENSITYPROPAGATION messages (e.g., in probability, law, or $L^p$). In fact, we have observed empirically that the situation where $\mu_{i \to j}$ converge but $m_{i \to j}$ do not converge (not even in distribution) is fairly common. It would be interesting to see if there is a variational interpretation for DENSITYPROPAGATION equations, similar to [13]. Notice also that Junction Tree style algorithms could also be used in conjunction with DP updates for the messages, as an instance of generalized distributive law [14].

## 4 Bounds on the density of states using tractable families

Using techniques such as DENSITYPROPAGATION, we can compute the density of states exactly for tractable families such as tree-structured graphical models. Let $p(x, \Theta^*)$ be a general (intractable) probabilistic model of interest, and let $\Theta_i$ be a family of tractable parameters (e.g., corresponding to trees) such that $\Theta^*$ is a convex combination of $\Theta_i$, as defined formally below and used previously

by Wainwright et al. [5, 6]. See below (Figure 1) for an example of a possible decomposition of a $2 \times 2$ Ising model into 2 tractable distributions. By computing the partition function or MAP estimates for the tree structured subproblems, Wainwright et al. showed that one can recover useful information about the original intractable problem, for instance by exploiting convexity of the log-partition function $\log Z(\Theta)$.

We present a way to exploit the decomposition idea to derive an upper bound on the density of states $n(E, \Theta^*)$ of the original intractable model, despite the fact that density of states is *not* a convex function of $\Theta^*$. The result below gives a point-by-point upper bound which, to the best of our knowledge, is the first bound of this kind for density of states. In the following, with some abuse of the notation, we denote $n(E, \Theta^*) = \sum_{x \in \mathcal{X}} \left( 1_{\{\Theta^* \cdot \phi(x) = E\}} \right)$ the function giving the number of configurations with energy $E$ (zero almost everywhere).

**Theorem 3.** *Let* $\Theta^* = \sum_{i=1}^{n} \gamma_i \Theta_i, \sum_{i=1}^{n} \gamma_i = 1$, *and* $y_n = E - \sum_{i=1}^{n-1} y_i$. *Then*

$$n(E, \Theta^*) \leq \int_{\mathbb{R}} \int_{\mathbb{R}} \dots \int_{\mathbb{R}} \min_{i=1}^{n} \left\{ n(y_i, \gamma_i \Theta_i) \right\} \mathrm{d}y_1 \mathrm{d}y_2 \dots \mathrm{d}y_{n-1}$$

*Proof.* From the definition of density of states and using $1_{\{\}}$ to denote the 0-1 indicator function,

$$n(E, \Theta^*) = \sum_{x \in \mathcal{X}} 1_{\{\Theta^* \phi(x) = E\}} = \sum_{x \in \mathcal{X}} 1_{\{(\sum_i \gamma_i \Theta_i) \phi(x) = E\}}$$

$$= \sum_{x \in \mathcal{X}} \int_{\mathbb{R}} \int_{\mathbb{R}} \dots \int_{\mathbb{R}} \left( \prod_{i=1}^{n} 1_{\{\gamma_i \Theta_i \phi(x) = y_i\}} \right) \mathrm{d}y_1 \mathrm{d}y_2 \dots \mathrm{d}y_{n-1} \quad \text{where } y_n = E - \sum_{i=1}^{n-1} y_i$$

$$= \int_{\mathbb{R}} \int_{\mathbb{R}} \dots \int_{\mathbb{R}} \sum_{x \in \mathcal{X}} \left( \prod_{i=1}^{n} 1_{\{\gamma_i \Theta_i \phi(x) = y_i\}} \right) \mathrm{d}y_1 \mathrm{d}y_2 \dots \mathrm{d}y_{n-1}$$

$$= \int_{\mathbb{R}} \int_{\mathbb{R}} \dots \int_{\mathbb{R}} \sum_{x \in \mathcal{X}} \left( \min_{i=1}^{n} \left\{ 1_{\{\gamma_i \Theta_i \phi(x) = y_i\}} \right\} \right) \mathrm{d}y_1 \mathrm{d}y_2 \dots \mathrm{d}y_{n-1}$$

$$\leq \int_{\mathbb{R}} \int_{\mathbb{R}} \dots \int_{\mathbb{R}} \min_{i=1}^{n} \left\{ \sum_{x \in \mathcal{X}} \left( 1_{\{\gamma_i \Theta_i \phi(x) = y_i\}} \right) \right\} \mathrm{d}y_1 \mathrm{d}y_2 \dots \mathrm{d}y_{n-1}$$

Observing that $\sum_{x \in \mathcal{X}} \left( 1_{\{\gamma_i \Theta_i \phi(x) = y_i\}} \right)$ is precisely $n(y_i, \gamma_i \Theta_i)$ finishes the proof. $\square$

## 5  Bounds on the partition function using n-dimensional matching

The density of states $n(E, \Theta^*)$ can be used to compute the partition function, since by definition $Z(\Theta^*) = ||n(E, \Theta^*) \exp(E)||_1$. We can therefore get an upper bound on $Z(\Theta^*)$ by integrating the point-by-point upper bound on $n(E, \Theta^*)$ from Theorem 3. This bound can be tighter than the known bound [6] obtained by applying Jensen's inequality to the log-partition function (which is convex), given by $\log Z(\Theta^*) \leq \sum_i \gamma_i \log Z(\Theta_i)$. For instance, consider a graphical model with weights that are large enough such that the density of states based sum defining $Z(\Theta^*)$ is dominated by the contribution of the highest-energy bucket. As a concrete example, consider the decomposition in Figure 1. As the edge weight $w$ ($w = 2$ in the figure) grows, the convexity-based bound will approximately equal the geometric average of $2 \exp(6w)$ and $8 \exp(2w)$, which is $4 \exp(4w)$. On the other hand, the bound based on Theorem 3 will approximately equal $\min\{2, 8\} \exp((2+6)w/2) = 2 \exp(4w)$. In general, the latter bound will always be strictly better for large enough $w$ unless the highest-energy bucket counts are identical across all $\Theta_i$.

While this is already promising, we can, in fact, obtain a much tighter bound by taking into account the interactions between different energy levels across any parameter decomposition, e.g., by enforcing the fact that there are a total of $|\mathcal{X}|$ configurations. For compactness, in the following let us define $y_i(x) = \exp(\Theta_i \cdot \phi(x))$ for any $x \in \mathcal{X}$ and $i = 1, \cdots, n$. Then,

$$Z(\Theta^*) = \sum_{x \in \mathcal{X}} \exp(\Theta^* \cdot \boldsymbol{\phi}(x)) = \sum_{x \in \mathcal{X}} \prod_i y_i(x)^{\gamma_i}$$

**Theorem 4.** *Let* $\Pi$ *be the (finite) set of all possible permutations of* $\mathcal{X}$. *Given* $\boldsymbol{\sigma} = (\sigma_1, \cdots, \sigma_n) \in \Pi^n$, *let* $Z(\Theta^*, \boldsymbol{\sigma}) = \sum_{x \in \mathcal{X}} \prod_i y_i(\sigma_i(x))^{\gamma_i}$. *Then,*

$$\min_{\boldsymbol{\sigma} \in \Pi^n} Z(\Theta^*, \boldsymbol{\sigma}) \leq Z(\Theta^*) \leq \max_{\boldsymbol{\sigma} \in \Pi^n} Z(\Theta^*, \boldsymbol{\sigma}) \tag{6}$$

---

**Algorithm 1** Greedy algorithm for the maximum matching (upper bound).

---
1: **while** there exists $E$ such that $n(E, \Theta_i) > 0$ **do**
2:    $E_{max}(\Theta_i) \leftarrow \max_E \{E | n(E, \Theta_i) > 0)\}$, for $i = 1, \cdots, n$
3:    $c' \leftarrow \min \{n(E_{max}(\Theta_1), \Theta_1), \cdots, n(E_{max}(\Theta_n), \Theta_n)\}$
4:    $u_b(\gamma_1 E_{max}(\Theta_1) + \cdots + \gamma_n E_{max}(\Theta_n), \Theta_1, \cdots, \Theta_n) \leftarrow c'$
5:    $n(E_{max}(\Theta_i), \Theta_i) \leftarrow n(E_{max}(\Theta_i), \Theta_i) - c'$, for $i = 1, \cdots, n$
6: **end while**

---

*Proof.* Let $\boldsymbol{\sigma_I} \in \Pi^n$ denote a collection of $n$ identity permutations. Then we have $Z(\Theta^*) = Z(\Theta^*, \boldsymbol{\sigma_I})$, which proves the upper and lower bounds in equation (6). $\qquad\square$

We can think of $\boldsymbol{\sigma} \in \Pi^n$ as an $n$-*dimensional matching* over the exponential size configuration space $\mathcal{X}$. For any $i, j$, $\sigma_i(x)$ matches with $\sigma_j(x)$, and $\boldsymbol{\sigma}(x)$ gives the corresponding hyper-edge. If we define the weight of each hyper-edge in the matching graph as $w(\boldsymbol{\sigma}(x)) = \prod_i y_i(\sigma_i(x))^{\gamma_i}$ then $Z(\Theta^*, \boldsymbol{\sigma}) = \sum_{x \in \mathcal{X}} w(\boldsymbol{\sigma}(x))$ corresponds to the weight of the matching represented by $\boldsymbol{\sigma}$. We can therefore think the bounds in equation (6) as given by a maximum and a minimum matching, respectively. Intuitively, the maximum matching corresponds to the case where the configurations in the high energy buckets of the densities happen to be the same configuration (matching), so that their energies are summed up.

## 5.1 Upper bound

The maximum matching $\max_{\boldsymbol{\sigma}} Z(\Theta^*, \boldsymbol{\sigma})$ (i.e., the upper bound on the partition function) can be computed using Algorithm 1. Algorithm 1 returns a distribution $u_b$ such that $\int u_b(E) \mathrm{d}E = |\mathcal{X}|$ and $\int u_b(E) \exp(E) \mathrm{d}E = \max_{\boldsymbol{\sigma}} Z(\Theta^*, \boldsymbol{\sigma})$. Notice however that $u_b(E)$ is not a valid point-by-point upper bound on the density $n(E, \Theta^*)$ of the original mode.

**Proposition 1.** *Algorithm 1 computes the maximum matching and its runtime is bounded by the total number of non-empty buckets $\sum_i |\{E | n(E, \Theta_i) > 0\}|$.*

*Proof.* The correctness of Algorithm 1 follows from observing that $\exp(E_1 + E_2) + \exp(E_1' + E_2') \geq \exp(E_1 + E_2') + \exp(E_1' + E_2)$ when $E_1 \geq E_1'$ and $E_2 \geq E_2'$. Intuitively, this means that for $n = 2$ parameters it is always optimal to connect the highest energy configurations, therefore the greedy method is optimal. This result can be generalized for $n > 2$ by induction. The runtime is proportional to the total number of buckets because we remove one bucket from at least one density at every iteration. $\qquad\square$

A key property of Algorithm 1 is that even though it defines a matching over an exponential number of configurations $|\mathcal{X}|$, its runtime proportional only to the total number of buckets, because it matches configurations in groups *at the bucket level*.

The following result shows that the value of the maximum matching is at least as tight as the bound provided by the convexity of the log-partition function, which is used for example by Tree Reweighted Belief Propagation (TRWBP) [6].

**Theorem 5.** *For any parameter decomposition $\sum_{i=1}^n \gamma_i \Theta_i = \Theta^*$, the upper bound given by the maximum matching in equation (6) and computed using Algorithm 1 is always at least as tight as the bound obtained using the convexity of the log-partition function.*

*Proof.* The bound obtained by applying Jensen's inequality to the log-partition function (which is convex), given by $\log Z(\Theta^*) \leq \sum_i \gamma_i \log Z(\Theta_i)$ [6], leads to the following geometric average bound $Z(\Theta^*) \leq \prod_i (\sum_x y_i(x))^{\gamma_i}$. Given any $n$ permutations of the configurations $\sigma_i : \mathcal{X} \to \mathcal{X}$ for $i = 1, \cdots, n$ (in particular, it holds for the one attaining the maximum matching value) we have

$$\sum_x \prod_i y_i(\sigma_i(x))^{\gamma_i} = || \prod_i y_i(\sigma_i(x))^{\gamma_i} ||_1 \leq \prod_i ||y_i(\sigma_i(x))^{\gamma_i}||_{1/\gamma_i} = \prod_i \left( \sum_x y_i(\sigma_i(x)) \right)^{\gamma_i}$$

where we used Generalized Holder inequality and the norm $|| \cdot ||_\ell$ indicates a sum over $\mathcal{X}$. $\qquad\square$

**Algorithm 2** Greedy algorithm for the minimum matching with $n = 2$ parameters (lower bound).

1: **while** there exists $E$ such that $n(E, \Theta_i) > 0$ **do**
2:     $E_{max}(\Theta_i) \leftarrow \max_E \{E | n(E, \Theta_i) > 0)\}$;   $E_{min}(\Theta_2) \leftarrow \min_E \{E | n(E, \Theta_2) > 0)\}$
3:     $c' \leftarrow \min \{n(E_{max}(\Theta_1), \Theta_1), n(E_{min}(\Theta_2), \Theta_2)\}$
4:     $l_b(\gamma_1 E_{max}(\Theta_1) + \gamma_2 E_{min}(\Theta_2), \Theta_1, \Theta_2) \leftarrow c'$
5:     $n(E_{max}(\Theta_1), \Theta_1) \leftarrow n(E_{max}(\Theta_1), \Theta_1) - c'$;   $n(E_{min}(\Theta_2), \Theta_2) \leftarrow n(E_{min}(\Theta_2), \Theta_2) - c'$
6: **end while**

## 5.2 Lower bound

We also provide Algorithm 2 to compute the minimum matching when there are $n = 2$ parameters. The proof of correctness is similar to that for Proposition 1.

**Proposition 2.** *For $n = 2$, Algorithm 2 computes the minimum matching and its runtime is bounded by the total number of non-empty buckets $\sum_i |\{E | n(E, \Theta_i) > 0\}|$.*

For the minimum matching case, the induction argument does not apply and the result does not extend to the case $n > 2$. For that case, we can obtain a weaker lower bound by applying Reverse Generalized Holder inequality [15], obtaining from a different perspective a bound previously derived in [16]. Specifically, let $s_1, \cdots, s_{n-1} < 0$ and $s_n$ such that $\sum \frac{1}{s_i} = 1$. We then have

$$\min_{\boldsymbol{\sigma}} Z(\Theta^*, \boldsymbol{\sigma}) = \sum_x \prod_i y_i(\sigma_{\min,i}(x))^{\gamma_i} = || \prod_i y_i(\sigma_{\min,i}(x))^{\gamma_i} ||_1 \geq \qquad (7)$$

$$\prod_i ||y_i(\sigma_{\min,i}(x))^{\gamma_i}||_{s_i} = \prod_i \left( \sum_x y_i(\sigma_{\min,i}(x))^{s_i \gamma_i} \right)^{\frac{1}{s_i}} = \prod_i \left( \sum_x y_i(x)^{s_i \gamma_i} \right)^{\frac{1}{s_i}}$$

Notice this result cannot be applied if $y_i(x) = 0$, i.e. there are factors assigning probability zero (hard constraints) in the probabilistic model.

# 6 Empirical evaluation

To evaluate the quality of the bounds, we consider an Ising model from statistical physics, where given a graph $(V, E)$, single node variables $x_s, s \in V$ are Bernoulli distributed $(x_s \in \{0, 1\})$, and the global random vector is distributed according to $p(x, \Theta) = \frac{1}{Z(\Theta)} \exp \left( \sum_{s \in V} \Theta_s x_s + \sum_{(i,j) \in E} \Theta_{ij} 1_{\{x_i = x_j\}} \right)$. Figure 1 shows a simple $2 \times 2$ grid Ising model with exponential parameter $\Theta^* = [0, 0, 0, 0, 1, 1, 1, 1]$ ($\Theta_s = 0$ and $\Theta_{ij} = 1$) decomposed as the convex sum of two parameters $\Theta_1$ and $\Theta_2$ corresponding to tractable distributions, i.e. $\Theta^* = (1/2)\Theta_1 + (1/2)\Theta_2$. The corresponding partition function is $Z(\Theta^*) = 2 + 12 \exp(2) + 2 \exp(4) \approx 199.86$. In panels 1(d) and 1(e) we report the corresponding density of states $n(E, \Theta_1)$ and $n(E, \Theta_2)$ as histograms. For instance, for the model corresponding to $\Theta_2$ there are only two global configurations (all variables positive and all negative) that give an energy of 6. It can be seen from the densities reported that $Z(\Theta_1) = 2 + 6 \exp(2) + 6 \exp(4) + 2 \exp(6) \approx 1180.8$, while $Z(\Theta_2) = 8 + 8 \exp(2) \approx 67.11$. The corresponding geometric average (obtained from the convexity of the log-partition function) is $\sqrt{(Z(\Theta_1))}\sqrt{(Z(\Theta_2))} \approx 281.50$. In panels 1(f) and 1(c) we show $u_b$ and $l_b$ computed using Algorithms 1 and 2, i.e. the solutions to the maximum and minimum matching problems, respectively. For instance, for the maximum matching case the 2 configurations with energy 6 from $n(E, \Theta_1)$ are matched with 2 of the 8 with energy 2 from $n(E, \Theta_2)$, giving an energy $6/2 + 2/2 = 4$. Notice that $u_b$ and $l_b$ are not valid bounds on individual densities of states themselves, but they nonetheless provide upper and lower bounds on the partition function as shown in the figure: $\approx 248.01$ and $134.27$, respectively. The bound (8) given by inverse Holder inequality with $s_1 = -1, s_2 = 1/2$ is $\approx 126.22$, while the mean field lower bound [4, 7] is $\approx 117.91$. In this case, the additional information provided by the density leads to tighter upper and lower bounds on the partition function.

In Figure 2 we report the upper bounds obtained for several types of Ising models (in all cases, $\Theta_s = 0$, i.e., there is no external field). In the two left plots, we consider a $N \times N$ square Ising model, once with attractive interactions ($\Theta_{ij} \in [0, w]$) and once with mixed interactions ($\Theta_{ij} \in [-w, w]$). In the two right plots, we use a complete graph (a clique) with $N = 15$ vertices. For each model, we compute the upper bound given by TRWBP (with edge appearance probabilities $\mu_e$ based on a

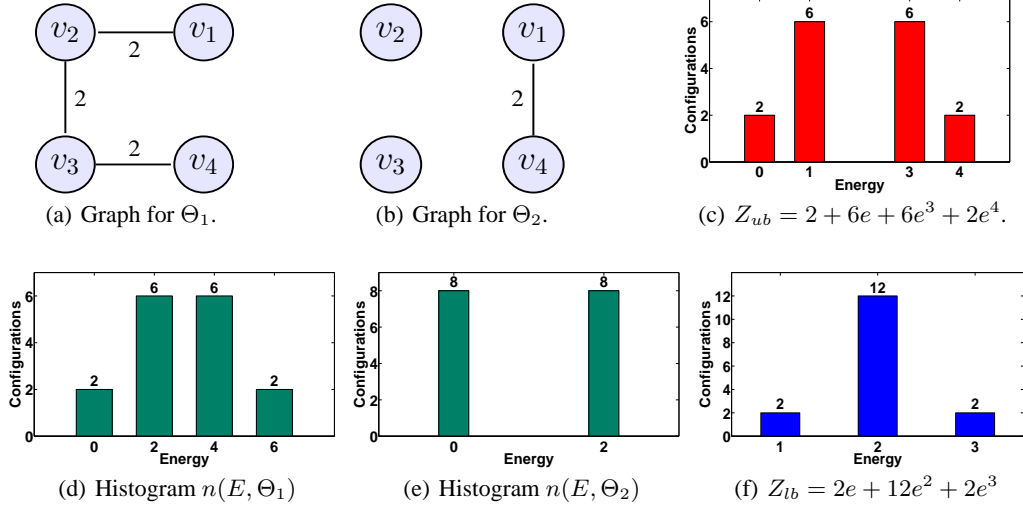

(a) Graph for $\Theta_1$.

(b) Graph for $\Theta_2$.

(c) $Z_{ub} = 2 + 6e + 6e^3 + 2e^4$.

(d) Histogram $n(E, \Theta_1)$

(e) Histogram $n(E, \Theta_2)$

(f) $Z_{lb} = 2e + 12e^2 + 2e^3$

Figure 1: Decomposition of a $2 \times 2$ Ising model, densities obtained with maximum and minimum matching algorithms, and the corresponding upper and lower bounds on $Z(\Theta^*)$.

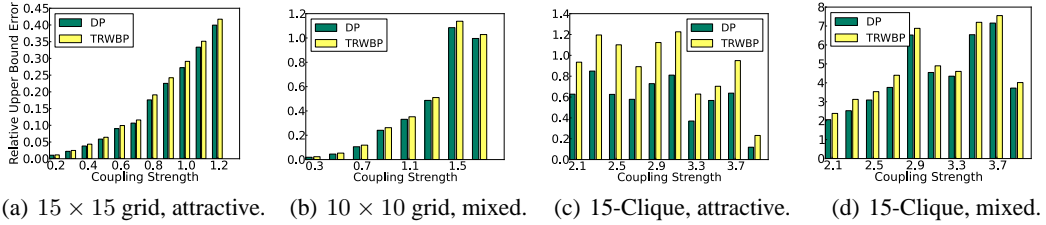

(a) $15 \times 15$ grid, attractive. (b) $10 \times 10$ grid, mixed. (c) 15-Clique, attractive. (d) 15-Clique, mixed.

Figure 2: Relative error of the upper bounds.

subset of 10 randomly selected spanning trees) and the mean-field bound using the implementations in libDAI [17]. We then compute the bound based on the maximum matching using the same set of spanning trees. For the grid case, we also use a combination of 2 spanning trees and compute the corresponding lower bound based on the minimum matching (notice it is not possible to cover all the edges in a clique with only 2 spanning tree). For each bound, we report the relative error, defined as $(\log(bound) - \log(Z)) / \log(Z)$, where $Z$ is the true partition function, computed using the junction tree method.

In these experiments, both our upper and lower bounds improve over the ones obtained with TR-WBP [6] and mean-field respectively. The lower bound based on minimum matching visually overlaps with the mean-field bound and is thus omitted from Figure 2. It is, however, strictly better, even if by a small amount. Notice that we might be able to get a better bound by choosing a different set of parameters $\Theta_i$ (which may be suboptimal for TRW-BP). By optimizing the parameters $s_i$ in the inverse Holder bound (8) using numerical optimization (BFGS and BOBYQA [18]), we were always able to obtain a lower bound at least as good as the one given by mean field.

## 7   Conclusions

We presented DENSITYPROPAGATION, a novel message passing algorithm for computing the density of states while exploiting the structure of the underlying graphical model. We showed that DENSITYPROPAGATION computes the exact density for tree structured graphical models and is a generalization of both Belief Propagation and Max-Product algorithms. We introduced a new family of bounds on the partition function based on $n$-dimensional matching and tree decomposition but without relying on convexity. The additional information provided by the density of states leads, both theoretically and empirically, to tighter bounds than known convexity-based ones.

## Footnotes

*Supported by NSF Expeditions in Computing award for Computational Sustainability (grant 0832782).

[1]Polynomial in the cardinality of the support, which could be exponential in $N$ in the worst case.

# References

[1] M.J. Wainwright and M.I. Jordan. Graphical models, exponential families, and variational inference. *Foundations and Trends in Machine Learning*, 1(1-2):1–305, 2008.

[2] S. Ermon, C. Gomes, A. Sabharwal, and B. Selman. Accelerated Adaptive Markov Chain for Partition Function Computation. *Neural Information Processing Systems*, 2011.

[3] F. Wang and DP Landau. Efficient, multiple-range random walk algorithm to calculate the density of states. *Physical Review Letters*, 86(10):2050–2053, 2001.

[4] M.J. Wainwright. *Stochastic processes on graphs with cycles: geometric and Variational approaches*. PhD thesis, Massachusetts Institute of Technology, 2002.

[5] M. Wainwright, T. Jaakkola, and A. Willsky. Exact map estimates by (hyper) tree agreement. *Advances in neural information processing systems*, pages 833–840, 2003.

[6] M.J. Wainwright. Tree-reweighted belief propagation algorithms and approximate ML estimation via pseudo-moment matching. In *AISTATS*, 2003.

[7] G. Parisi and R. Shankar. Statistical field theory. *Physics Today*, 41:110, 1988.

[8] L.D. Brown. *Fundamentals of statistical exponential families: with applications in statistical decision theory*. Institute of Mathematical Statistics, 1986.

[9] M. Richardson and P. Domingos. Markov logic networks. *Machine Learning*, 62(1):107–136, 2006.

[10] Y. Weiss, C. Yanover, and T. Meltzer. MAP estimation, linear programming and belief propagation with convex free energies. In *Uncertainty in Artificial Intelligence*, 2007.

[11] T. Hazan and A. Shashua. Norm-product belief propagation: Primal-dual message-passing for approximate inference. *Information Theory, IEEE Transactions on*, 56(12):6294–6316, 2010.

[12] K.P. Murphy, Y. Weiss, and M.I. Jordan. Loopy belief propagation for approximate inference: An empirical study. In *Proceedings of the Fifteenth conference on Uncertainty in artificial intelligence*, pages 467–475. Morgan Kaufmann Publishers Inc., 1999.

[13] J.S. Yedidia, W.T. Freeman, and Y. Weiss. Understanding belief propagation and its generalizations. *Exploring artificial intelligence in the new millennium*, 8:236–239, 2003.

[14] S.M. Aji and R.J. McEliece. The generalized distributive law. *Information Theory, IEEE Transactions on*, 46(2):325–343, 2000.

[15] W.S. Cheung. Generalizations of Hölders inequality. *International Journal of Mathematics and Mathematical Sciences*, 26:7–10, 2001.

[16] Qiang Liu and Alexander Ihler. Negative tree reweighted belief propagation. In *Proceedings of the Twenty-Sixth Conference Annual Conference on Uncertainty in Artificial Intelligence (UAI-10)*, pages 332–339, Corvallis, Oregon, 2010. AUAI Press.

[17] J.M. Mooij. libDAI: A free and open source c++ library for discrete approximate inference in graphical models. *The Journal of Machine Learning Research*, 11:2169–2173, 2010.

[18] M.J.D. Powell. The BOBYQA algorithm for bound constrained optimization without derivatives. *University of Cambridge Technical Report*, 2009.

